# Approximate Dynamic Programming via Linear Programming

**Daniela P. de Farias**
Department of Management Science and Engineering
Stanford University
Stanford, CA 94305
*pucci@stanford.edu*

**Benjamin Van Roy**
Department of Management Science and Engineering
Stanford University
Stanford, CA 94305
*bvr@stanford.edu*

## Abstract

The curse of dimensionality gives rise to prohibitive computational requirements that render infeasible the exact solution of large–scale stochastic control problems. We study an efficient method based on linear programming for approximating solutions to such problems. The approach "fits" a linear combination of pre–selected basis functions to the dynamic programming cost–to–go function. We develop bounds on the approximation error and present experimental results in the domain of queueing network control, providing empirical support for the methodology.

## 1 Introduction

Dynamic programming offers a unified approach to solving problems of stochastic control. Central to the methodology is the cost–to–go function, which can obtained via solving Bellman's equation. The domain of the cost–to–go function is the state space of the system to be controlled, and dynamic programming algorithms compute and store a table consisting of one cost–to–go value per state. Unfortunately, the size of a state space typically grows exponentially in the number of state variables. Known as the *curse of dimensionality*, this phenomenon renders dynamic programming intractable in the face of problems of practical scale.

One approach to dealing with this difficulty is to generate an approximation within a parameterized class of functions, in a spirit similar to that of statistical regression. The focus of this paper is on linearly parameterized functions: one tries to approximate the cost–to–go function $J^*$ by a linear combination of prespecified basis functions. Note that this scheme depends on two important preconditions for the development of an effective approximation. First, we need to choose basis functions

that can closely approximate the desired cost-to-go function. In this respect, a suitable choice requires some practical experience or theoretical analysis that provides rough information on the shape of the function to be approximated. "Regularities" associated with the function, for example, can guide the choice of representation. Second, we need an efficient algorithm that computes an appropriate linear combination.

The algorithm we study is based on a linear programming formulation, originally proposed by Schweitzer and Seidman [5], that generalizes the linear programming approach to exact dynamic programming, originally introduced by Manne [4]. We present an error bound that characterizes the quality of approximations produced by the linear programming approach. The error is characterized in relative terms, compared against the "best possible" approximation of the optimal cost-to-go function given the selection of basis functions. This is the first such error bound for any algorithm that approximates cost–to–go functions of general stochastic control problems by computing weights for arbitrary collections of basis functions.

## 2 Stochastic control and linear programming

We consider discrete–time stochastic control problems involving a finite state space $\mathcal{S}$ of cardinality $|\mathcal{S}| = N$. For each state $x \in \mathcal{S}$, there is a finite set of available actions $\mathcal{A}_x$. Taking action $a \in \mathcal{A}_x$ when the current state is $x$ incurs cost $g_a(x)$. State transition probabilities $p_a(x, y)$ represent, for each pair $(x, y)$ of states and each action $a \in \mathcal{A}_x$, the probability that the next state will be $y$ given that the current state is $x$ and the current action is $a \in \mathcal{A}_x$.

A *policy* $u$ is a mapping from states to actions. Given a policy $u$, the dynamics of the system follow a Markov chain with transition probabilities $p_{u(x)}(x, y)$. For each policy $u$, we define a transition matrix $P_u$ whose $(x, y)$th entry is $p_{u(x)}(x, y)$.

The problem of stochastic control amounts to selection of a policy that optimizes a given criterion. In this paper, we will employ as an optimality criterion infinite–horizon discounted cost of the form

$$J_u(x) = \mathrm{E}\left[\sum_{t=0}^{\infty} \alpha^t g_u(x_t) \Big| x_0 = x\right],$$

where $g_u(x)$ is used as shorthand for $g_{u(x)}(x)$ and the discount factor $\alpha \in (0, 1)$ reflects inter–temporal preferences. Optimality is attained by any policy that is greedy with respect to the optimal cost-to-go function $J^*(x) = \min_u J_u(x)$ (a policy $u$ is called greedy with respect to $J$ if $T_u J = TJ$).

Let us define operators $T_u$ and $T$ by $T_u J = g_u + \alpha P_u J$ and $TJ = \min_u (g_u + \alpha P_u J)$. The optimal cost-to-go function solves uniquely Bellman's equation $J = TJ$. Dynamic programming offers a number of approaches to solving this equation; one of particular relevance to our paper makes use of linear programming, as we will now discuss. Consider the problem

$$\max \quad c'J \tag{1}$$
$$\text{s.t.} \quad TJ \geq J,$$

where $c$ is a vector with positive components, which we will refer to as *state-relevance weights*. It can be shown that any feasible $J$ satisfies $J \leq J^*$. It follows that, for any set of positive weights $c$, $J^*$ is the unique solution to (1).

Note that each constraint $(TJ)(x) \geq J(x)$ is equivalent to a set of constraints $g_a(x) + \alpha \sum_{y \in \mathcal{S}} p_a(x, y) J(y) \geq J(x)$, $\forall a \in \mathcal{A}_x$, so that the optimization problem (1) can be represented as an LP, which we refer to as the *exact LP*.

As mentioned in the introduction, state spaces for practical problems are enormous due to the curse of dimensionality. Consequently, the linear program of interest involves prohibitively large numbers of variables and constraints. The approximation algorithm we study reduces dramatically the number of variables.

Let us now introduce the linear programming approach to approximate dynamic programming. Given pre–selected basis functions $\phi_1, \ldots, \phi_K$, define a matrix $\Phi = [\ \phi_1 \ \cdots \ \phi_K\ ]$. With an aim of computing a weight vector $\tilde{r} \in \Re^K$ such that $\Phi\tilde{r}$ is a close approximation to $J^*$, one might pose the following optimization problem:

$$\max \quad c'\Phi r \qquad (2)$$
$$\text{s.t.} \quad T\Phi r \geq \Phi r.$$

Given a solution $\tilde{r}$, one might then hope to generate near–optimal decisions by using a policy that is greedy with respect to $\Phi\tilde{r}$.

As with the case of exact dynamic programming, the optimization problem (2) can be recast as a linear program. We will refer to this problem as the *approximate LP*. Note that, though the number of variables is reduced to $K$, the number of constraints remains as large as in the exact LP. Fortunately, we expect that most of the constraints will become irrelevant, and solutions to the linear program can be approximated efficiently, as demonstrated in [3].

## 3 Error Bounds for the Approximate LP

When the optimal cost–to–go function lies within the span of the basis functions, solution of the approximate LP yields the exact optimal cost–to–go function. Unfortunately, it is difficult in practice to select a set of basis functions that contains the optimal cost–to–go function within its span. Instead, basis functions must be based on heuristics and simplified analyses. One can only hope that the span comes close to the desired cost–to–go function.

For the approximate LP to be useful, it should deliver good approximations when the cost–to–go function is near the span of selected basis functions. In this section, we present a bound that ensure desirable results of this kind.

To set the stage for development of an error bound, let us establish some notation. First, we introduce the weighted norms, defined by

$$\|J\|_{1,\gamma} = \sum_{x \in \mathcal{S}} \gamma(x)|J(x)|, \ \|J\|_{\infty,\gamma} = \max_{x \in \mathcal{S}} \gamma(x)|J(x)|,$$

for any $\gamma : \mathcal{S} \mapsto \Re^+$. Note that both norms allow for uneven weighting of errors across the state space.

We also introduce an operator $H$, defined by

$$(HV)(x) = \max_{a \in \mathcal{A}_x} \sum_y P_a(x,y)V(y),$$

for all $V : \mathcal{S} \mapsto \Re$. For any $V$, $(HV)(x)$ represents the maximum expected value of $V(y)$ if the current state is $x$ and $y$ is a random variable representing the next state. Based on this operator, we define a scalar

$$k_V = \max_x \frac{V(x)}{V(x) - \alpha(HV)(x)}, \qquad (3)$$

for each $V : \mathcal{S} \mapsto \Re$.

We interpret the argument $V$ of $H$ as a "Lyapunov function," while we view $k_V$ as a "Lyapunov stability factor," in a sense that we will now explain. In the upcoming theorem, we will only be concerned with functions $V$ that are positive and that make $k_V$ nonnegative. Also, our error bound for the approximate LP will grow proportionately with $k_V$, and we therefore want $k_V$ to be small. At a minimum, $k_V$ should be finite, which translates to a condition

$$\alpha(HV)(x) < V(x), \qquad \forall x \in \mathcal{S}. \qquad (4)$$

If $\alpha$ were equal to 1, this would look like a Lyapunov stability condition: the maximum expected value $(HV)(x)$ at the next time step must be less than the current value $V(x)$. In general, $\alpha$ is less than 1, and this introduces some slack in the condition. Note also that $k_V$ becomes smaller as the $(HV)(x)$'s become small relative to the $V(x)$'s. Hence, $k_V$ conveys a degree of "stability," with smaller values representing stronger stability.

We are now ready to state our main result. For any given function $V$ mapping $\mathcal{S}$ to positive reals, we use $1/V$ as shorthand for a function $x \mapsto 1/V(x)$.

**Theorem 3.1** *[2] Let $\tilde{r}$ be a solution of the approximate LP. Then, for any $v \in \Re^K$ such that $(\Phi v)(x) > 0$ for all $x \in \mathcal{S}$ and $\alpha H \Phi v < \Phi v$,*

$$\|J^* - \Phi \tilde{r}\|_{1,c} \leq 2k_{\Phi v}(c'\Phi v) \min_r \|J^* - \Phi r\|_{\infty, 1/\Phi v}. \qquad (5)$$

A proof of Theorem 3.1 can be found in the long version of this paper [2].

We highlight some implications of Theorem 3.1. First, the error bound (5) tells that the the approximation error yielded by the approximate LP is proportional to the error associated with the best possible approximation relative to a certain norm $\|\cdot\|_{1,1/\Phi v}$. Hence we expect that the approximate LP will have reasonable behavior – if the choice of basis functions is appropriate, the approximate LP should yield a relatively good approximation to the cost-to-go function, as long as the constants $k_{\Phi v}$ and $c'\Phi v$ remain small.

Note that on the left-hand side of (5), we measure the approximation error with the weighted norm $\|\cdot\|_{1,c}$. Recall that the weight vector $c$ appears in objective function of the approximate LP (2) and must be chosen. In approximating the solution to a given stochastic control problem, it seems sensible to weight more heavily portions of the state space that are visited frequently, so that accuracy will be emphasized in such regions. As discussed in [2], it seems reasonable that the weight vector $c$ should be chosen to reflect the relative importance of each state.

Finally, note that the Lyapunov function $\Phi v$ plays a central role in the bound of Theorem 3.1. Its choice influences three terms on the right–hand–side of the bound:

1. the error $\min_r \|J^* - \Phi r\|_{\infty, 1/\Phi v}$;

2. the Lyapunov stability factor $k_{\Phi v}$;

3. the inner product $c'\Phi v$ with the state–relevance weights.

An appropriately chosen Lyapunov function should make all three of these terms relatively small. Furthermore, for the bound to be useful in practical contexts, these terms should not grow much with problem size. We now illustrate with an application in queueing problems how a suitable Lyapunov function could be found and show how these terms scale with problem size.

### 3.1 Example: A Queueing Network

Consider a single reentrant line with $d$ queues and finite buffers of size B. We assume that exogenous arrivals occur at queue 1 with probability $p < 1/2$. The state $x \in \Re^d$ indicates the number of jobs in each queue. The cost per stage incurred at state $x$ is given by

$$g(x) = \frac{|x|}{d} = \frac{1}{d} \sum_{i=1}^{d} x_i,$$

the average number of jobs per queue.

As discussed in [2], under certain stability assumptions we expect that the optimal cost-to-go function should satisfy

$$0 \le J^*(x) \le \frac{\rho_2}{d} x'x + \frac{\rho_1}{d} e'x + \rho_0,$$

for some positive scalars $\rho_0$, $\rho_1$ and $\rho_2$ independent of $d$. We consider a Lyapunov function $V(x) = \frac{1}{d} x'x + C$ for some constant $C > 0$, which implies

$$
\begin{aligned}
\min_r \|J^* - \Phi r\|_{\infty,1/V} &\le \|J^*\|_{\infty,1/V} \\
&\le \max_{x \ge 0} \frac{\rho_2 x'x + \rho_1 e'x + d\rho_0}{x'x + dC} \\
&\le \rho_2 + \rho_1 + \frac{\rho_0}{C},
\end{aligned}
$$

and the above bound is independent of the number of queues in the system.

Now let us study $k_V$. We have

$$
\begin{aligned}
\alpha(HV)(x) &\le \alpha \left[ p\left( \frac{1}{d}x'x + \frac{2x_1 + 1}{d} + C \right) + (1-p)\left( \frac{1}{d}x'x + C \right) \right] \\
&\le V(x) \left( \alpha + \alpha p \frac{2x_1 + 1}{x_1^2 + dC} \right),
\end{aligned}
$$

and it is clear that, for $C$ sufficiently large and independent of $d$, there is a $\beta < 1$ independent of $d$ such that $\alpha HV \le \beta V$, and therefore $k_V \le \frac{1}{1-\beta}$.

Finally, let us consider $c'V$. Discussion presented in [2] suggests that one might want to choose $c$ so as to reflect the stationary state distribution. We expect that under some stability assumptions, the tail of the stationary state distribution will have an upper bound with geometric decay [1]. Therefore we let $c(x) = \left( \frac{1-\rho}{1-\rho^{B+1}} \right)^d \rho^{|x|}$, for some $0 < \rho < 1$. In this case, $c$ is equivalent to the conditional joint distribution of $d$ independent and identically distributed geometric random variables conditioned on the event that they are less than $B + 1$, and we have

$$c'V = E\left[ \frac{1}{d} \sum_{i=1}^{d} X_i^2 + C \ \Big| \ X_i < B+1, i = 1,...,d \right] < 2\frac{\rho^2}{(1-\rho)^2} + \frac{\rho}{1-\rho} + C,$$

where $X_i, i = 1,...,d$ are identically distributed geometric random variables with parameter $1 - \rho$. It follows that $c'V$ is uniformly bounded over the number of queues.

This example shows that the terms involved in the error bound (5) are uniformly bounded both in the number of states in the system and in the number of state variables, hence the behavior of the approximate LP does not deteriorate as the problem size increases.

We finally present a numerical experiment to further illustrate the performance of the approximate LP.

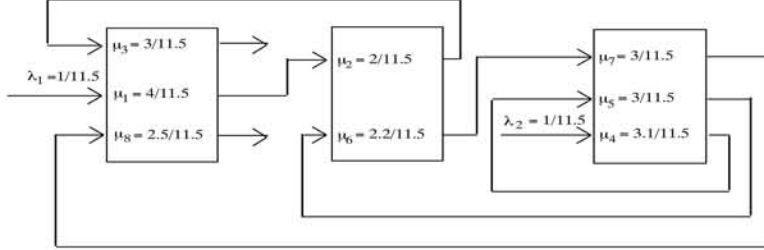

Figure 1: System for Example 3.2.

| Policy | ALP($\rho = 0.9$) | LBFS | FIFO | LONG |
|---|---|---|---|---|
| Average Cost | 136.7 | 153.3 | 163.3 | 168.3 |

Table 1: Average number of jobs after 50,000,000 simulation steps

## 3.2 An Eight-Dimensional Queueing Network

We consider a queueing network with eight queues. The system is depicted in Figure 1, with arrival ($\lambda_i, i = 1, 2$) and departure ($\mu_i, i = 1, ..., 8$) probabilities indicated.

The state $x \in \Re^8$ represents the number of jobs in each queue. The cost-per-state is $g(x) = |x|$, and the discount factor $\alpha$ is 0.995. Actions $a \in \{0, 1\}^8$ indicate which queues are being served; $a_i = 1$ iff a job from queue $i$ is being processed. We consider only non-iddling policies and, at each time step, a server processes jobs from one of its queues exclusively.

We choose $c$ of the form $c(x) = (1 - \rho)^8 \rho^{|x|}$. The basis functions are chosen to span all polynomials in $x$ of degree 2; therefore, the approximate LP has 47 variables. Constraints $(T\Phi r)(x) \geq (\Phi r)(x)$ for the approximate LP are generated by sampling 5000 states according to the distribution associated with $c$. Experiments were performed for $\rho = 0.85, 0.9$ and $0.95$, and $\rho = 0.9$ yielded the policy with smallest average cost.

We compared the performance of the policy yielded by the approximate LP (ALP) with that of first-in-first-out (FIFO), last-buffer-first-serve (LBFS)[1] and a policy that serves the longest queue in each server (LONG). The average number of jobs in the system for each policy was estimated by simulation. Results are shown in Table 1. The policy generated by the approximate LP performs significantly better than each of the heuristics, yielding more than 10% improvement over LBFS, the second best policy. We expect that even better results could be obtained by refining the choice of basis functions and state-relevance weights.

## 4 Closing Remarks and Open Issues

In this paper we studied the linear programming approach to approximate dynamic programming for stochastic control problems as a means of alleviating the curse of

dimensionality. We provided an error bound based on certain assumptions on the basis functions. The bounds were shown to be uniformly bounded in the number of states and state variables in certain queueing problems.

Several questions remain open and are the object of future investigation: Can the state-relevance weights in the objective function be chosen in some adaptive way? Can we add robustness to the approximate LP algorithm to account for errors in the estimation of costs and transition probabilities, i.e., design an alternative LP with meaningful performance bounds when problem parameters are just known to be in a certain range? How do our results extend to the average cost case? How do our results extend to the infinite-state case? How does the quality of the approximate value function, measure by the weighted $L_1$ norm, translate into actual performance of the associated greedy policy?

## Acknowledgements

This research was supported by NSF CAREER Grant ECS-9985229, by the ONR under Grant MURI N00014-00-1-0637, and by an IBM Research Fellowship.

## Footnotes

[1]LBFS serves the job that is closest to leaving the system; for example, if there are jobs in queue 2 and in queue 6, a job from queue 2 is processed since it will leave the system after going through only one more queue, whereas the job from queue 6 will still have to go through two more queues. We also choose to assign higher priority to queue 8 than to queue 3 since queue 8 has higher departure probability.

## References

[1] Bertsimas, D., Gamarnik, D. & Tsitsiklis, J., "Performance of Multiclass Markovian Queueing Networks via Piecewise Linear Lyapunov Functions," submitted to *Annals of Applied Probability,* 2000.

[2] de Farias, D.P. & Van Roy, B., "The Linear Programming Approach to Approximate Dynamic Programming," submitted to publication, 2001.

[3] de Farias, D.P. & Van Roy, B., "On Constraint Sampling for Approximate Linear Programming,", submitted to publication, 2001.

[4] Manne, A.S., "Linear Programming and Sequential Decisions," *Management Science 6*, No. 3, pp. 259-267, 1960.

[5] Schweitzer, P. & Seidmann, A., "Generalized Polynomial Approximations in Markovian Decision Processes," *Journal of Mathematical Analysis and Applications 110*, pp. 568-582, 1985.
